# Policy Search for Motor Primitives in Robotics

**Jens Kober, Jan Peters**
Max Planck Institute for Biological Cybernetics
Spemannstr. 38
72076 Tübingen, Germany
{jens.kober,jan.peters}@tuebingen.mpg.de

## Abstract

Many motor skills in humanoid robotics can be learned using parametrized motor primitives as done in imitation learning. However, most interesting motor learning problems are high-dimensional reinforcement learning problems often beyond the reach of current methods. In this paper, we extend previous work on policy learning from the immediate reward case to episodic reinforcement learning. We show that this results in a general, common framework also connected to policy gradient methods and yielding a novel algorithm for policy learning that is particularly well-suited for dynamic motor primitives. The resulting algorithm is an EM-inspired algorithm applicable to complex motor learning tasks. We compare this algorithm to several well-known parametrized policy search methods and show that it outperforms them. We apply it in the context of motor learning and show that it can learn a complex Ball-in-a-Cup task using a real Barrett WAM™ robot arm.

## 1 Introduction

Policy search, also known as policy learning, has become an accepted alternative of value function-based reinforcement learning [2]. In high-dimensional domains with continuous states and actions, such as robotics, this approach has previously proven successful as it allows the usage of domain-appropriate pre-structured policies, the straightforward integration of a teacher's presentation as well as fast online learning [2, 3, 10, 18, 5, 6, 4]. In this paper, we will extend the previous work in [17, 18] from the immediate reward case to episodic reinforcement learning and show how it relates to policy gradient methods [7, 8, 11, 10]. Despite that many real-world motor learning tasks are essentially episodic [14], episodic reinforcement learning [1] is a largely undersubscribed topic. The resulting framework allows us to derive a new algorithm called Policy Learning by Weighting Exploration with the Returns (PoWER) which is particularly well-suited for learning of trial-based tasks in motor control. We are especially interested in a particular kind of motor control policies also known as dynamic motor primitives [22, 23]. In this approach, dynamical systems are being used in order to encode a policy, i.e., we have a special kind of parametrized policy which is well-suited for robotics problems.

We show that the presented algorithm works well when employed in the context of learning dynamic motor primitives in four different settings, i.e., the two benchmark problems from [10], the Underactuated Swing-Up [21] and the complex task of Ball-in-a-Cup [24, 20]. Both the Underactuated Swing-Up as well as the Ball-in-a-Cup are achieved on a real Barrett WAM™ robot arm. Please also refer to the video on the first author's website. Looking at these tasks from a human motor learning perspective, we have a human acting as teacher presenting an example for imitation learning and, subsequently, the policy will be improved by reinforcement learning. Since such tasks are inherently single-stroke movements, we focus on the special class of episodic reinforcement learning. In our experiments, we show how a presented movement is recorded using kinesthetic teach-in and, subsequently, how a Barrett WAM™ robot arm is learning the behavior by a combination of imitation and reinforcement learning.

## 2 Policy Search for Parameterized Motor Primitives

Our goal is to find reinforcement learning techniques that can be applied to a special kind of pre-structured parametrized policies called *motor primitives* [22, 23], in the context of learning high-dimensional motor control tasks. In order to do so, we first discuss our problem in the general context of reinforcement learning and introduce the required notation in Section 2.1. Using a generalization of the approach in [17, 18], we derive a new EM-inspired algorithm called Policy Learning by Weighting Exploration with the Returns (PoWER) in Section 2.3 and show how the general framework is related to policy gradients methods in 2.2. [12] extends the [17] algorithm to episodic reinforcement learning for discrete states; we use continuous states. Subsequently, we discuss how we can turn the parametrized motor primitives [22, 23] into explorative [19], stochastic policies.

### 2.1 Problem Statement & Notation

In this paper, we treat motor primitive learning problems in the framework of reinforcement learning with a strong focus on the episodic case [1]. We assume that at time $t$ there is an actor in a state $\mathbf{s}_t$ and chooses an appropriate action $\mathbf{a}_t$ according to a stochastic policy $\pi(\mathbf{a}_t|\mathbf{s}_t, t)$. Such a policy is a probability distribution over actions given the current state. The stochastic formulation allows a natural incorporation of exploration and, in the case of hidden state variables, the optimal time-invariant policy has been shown to be stochastic [8]. Upon the completion of the action, the actor transfers to a state $\mathbf{s}_{t+1}$ and receives a reward $r_t$. As we are interested in learning complex motor tasks consisting of a single stroke [23], we focus on finite horizons of length $T$ with episodic restarts [1] and learn the optimal parametrized, stochastic policy for such reinforcement learning problems. We assume an explorative version of the dynamic motor primitives [22, 23] as parametrized policy $\pi$ with parameters $\boldsymbol{\theta} \in \mathbb{R}^n$. However, in this section, we will keep most derivations sufficiently general that they would transfer to various other parametrized policies. The general goal in reinforcement learning is to optimize the *expected return* of the policy $\pi$ with parameters $\boldsymbol{\theta}$ defined by

$$J(\boldsymbol{\theta}) = \int_{\mathbb{T}} p(\boldsymbol{\tau})R(\boldsymbol{\tau})d\boldsymbol{\tau}, \qquad (1)$$

where $\mathbb{T}$ is the set of all possible paths, rollout $\boldsymbol{\tau} = [\mathbf{s}_{1:T+1}, \mathbf{a}_{1:T}]$ (also called episode or trial) denotes a path of states $\mathbf{s}_{1:T+1} = [\mathbf{s}_1, \mathbf{s}_2, \ldots, \mathbf{s}_{T+1}]$ and actions $\mathbf{a}_{1:T} = [\mathbf{a}_1, \mathbf{a}_2, \ldots, \mathbf{a}_T]$. The probability of rollout $\boldsymbol{\tau}$ is denoted by $p(\boldsymbol{\tau})$ while $R(\boldsymbol{\tau})$ refers to its return. Using the standard assumptions of Markovness and additive accumulated rewards, we can write

$$p(\boldsymbol{\tau}) = p(\mathbf{s}_1)\prod_{t=1}^{T}p(\mathbf{s}_{t+1}|\mathbf{s}_t, \mathbf{a}_t)\pi(\mathbf{a}_t|\mathbf{s}_t, t), \qquad R(\boldsymbol{\tau}) = T^{-1}\sum_{t=1}^{T}r(\mathbf{s}_t, \mathbf{a}_t, \mathbf{s}_{t+1}, t), \qquad (2)$$

where $p(\mathbf{s}_1)$ denotes the initial state distribution, $p(\mathbf{s}_{t+1}|\mathbf{s}_t, \mathbf{a}_t)$ the next state distribution conditioned on last state and action, and $r(\mathbf{s}_t, \mathbf{a}_t, \mathbf{s}_{t+1}, t)$ denotes the immediate reward.

While episodic Reinforcement Learning (RL) problems with finite horizons are common in motor control, few methods exist in the RL literature, e.g., Episodic REINFORCE [7], the Episodic Natural Actor Critic eNAC [10] and model-based methods using differential-dynamic programming [21]. Nevertheless, in the analytically tractable cases, it has been studied deeply in the optimal control community where it is well-known that for a finite horizon problem, the optimal solution is non-stationary [15] and, in general, cannot be represented by a time-independent policy. The motor primitives based on dynamical systems [22, 23] are a particular type of time-variant policy representation as they have an internal phase which corresponds to a clock with additional flexibility (e.g., for incorporating coupling effects, perceptual influences, etc.), thus, they can represent optimal solutions for finite horizons. We embed this internal clock or movement phase into our state and, thus, from optimal control perspective have ensured that the optimal solution can be represented.

### 2.2 Episodic Policy Learning

In this section, we discuss episodic reinforcement learning in policy space which we will refer to as Episodic Policy Learning. For doing so, we first discuss the lower bound on the expected return suggested in [17] for guaranteeing that policy update steps are improvements. In [17, 18] only the immediate reward case is being discussed, we extend their framework to episodic reinforcement learning and, subsequently, derive a general update rule which yields the policy gradient theorem [8], a generalization of the reward-weighted regression [18] as well as the novel Policy learning by Weighting Exploration with the Returns (PoWER) algorithm.

#### 2.2.1 Bounds on Policy Improvements

Unlike in reinforcement learning, other machine learning branches have focused on optimizing lower bounds, e.g., resulting in expectation-maximization (EM) algorithms [16]. The reasons for this preference apply in policy learning: if the lower bound also becomes an equality for the sampling policy,

we can guarantee that the policy will be improved by optimizing the lower bound. Surprisingly, results from supervised learning can be transferred with ease. For doing so, we follow the scenario suggested in [17], i.e., generate rollouts $\tau$ using the current policy with parameters $\boldsymbol{\theta}$ which we weight with the returns $R(\tau)$ and subsequently match it with a new policy parametrized by $\boldsymbol{\theta}'$. This matching of the success-weighted path distribution is equivalent to minimizing the Kullback-Leibler divergence $D\left(p_{\boldsymbol{\theta}'}(\tau)\|p_{\boldsymbol{\theta}}(\tau)R(\tau)\right)$ between the new path distribution $p_{\boldsymbol{\theta}'}(\tau)$ and the reward-weighted previous one $p_{\boldsymbol{\theta}}(\tau)R(\tau)$. As shown in [17, 18], this results in a lower bound on the expected return using Jensen's inequality and the concavity of the logarithm, i.e.,

$$\log J(\boldsymbol{\theta}') = \log \int_{\mathbb{T}} \frac{p_{\boldsymbol{\theta}}(\tau)}{p_{\boldsymbol{\theta}}(\tau)} p_{\boldsymbol{\theta}'}(\tau) R(\tau) d\tau \geq \int_{\mathbb{T}} p_{\boldsymbol{\theta}}(\tau) R(\tau) \log \frac{p_{\boldsymbol{\theta}'}(\tau)}{p_{\boldsymbol{\theta}}(\tau)} d\tau + \text{const}, \quad (3)$$

$$\propto -D\left(p_{\boldsymbol{\theta}}(\tau) R(\tau)\|p_{\boldsymbol{\theta}'}(\tau)\right) = L_{\boldsymbol{\theta}}(\boldsymbol{\theta}'), \quad (4)$$

where $D\left(p(\tau)\|q(\tau)\right) = \int p(\tau) \log(p(\tau)/q(\tau)) d\tau$ is the Kullback-Leibler divergence which is considered a natural distance measure between probability distributions, and the constant is needed for tightness of the bound. Note that $p_{\boldsymbol{\theta}}(\tau) R(\tau)$ is an improper probability distribution as pointed out in [17]. The policy improvement step is equivalent to maximizing the lower bound on the expected return $L_{\boldsymbol{\theta}}(\boldsymbol{\theta}')$ and we show how it relates to previous policy learning methods.

### 2.2.2 Resulting Policy Updates

In the following part, we will discuss three different policy updates which directly result from Section 2.2.1. First, we show that policy gradients [7, 8, 11, 10] can be derived from the lower bound $L_{\boldsymbol{\theta}}(\boldsymbol{\theta}')$ (as was to be expected from supervised learning, see [13]). Subsequently, we show that natural policy gradients can be seen as an additional constraint regularizing the change in the path distribution resulting from a policy update when improving the policy incrementally. Finally, we will show how expectation-maximization (EM) algorithms for policy learning can be generated.

**Policy Gradients.** When differentiating the function $L_{\boldsymbol{\theta}}(\boldsymbol{\theta}')$ that defines the lower bound on the expected return, we directly obtain

$$\partial_{\boldsymbol{\theta}'} L_{\boldsymbol{\theta}}(\boldsymbol{\theta}') = \int_{\mathbb{T}} p_{\boldsymbol{\theta}}(\tau) R(\tau) \partial_{\boldsymbol{\theta}'} \log p_{\boldsymbol{\theta}'}(\tau) d\tau, \quad (5)$$

where $\mathbb{T}$ is the set of all possible paths and $\partial_{\boldsymbol{\theta}'} \log p_{\boldsymbol{\theta}'}(\tau) = \sum_{t=1}^{T} \partial_{\boldsymbol{\theta}'} \log \pi(\mathbf{a}_t|\mathbf{s}_t, t)$ denotes the log-derivative of the path distribution. As this log-derivative only depends on the policy, we can estimate a gradient from rollouts without having a model by simply replacing the expectation by a sum; when $\boldsymbol{\theta}'$ is close to $\boldsymbol{\theta}$, we have the policy gradient estimator which is widely known as Episodic REINFORCE [7], i.e., we have $\lim_{\boldsymbol{\theta}' \to \boldsymbol{\theta}} \partial_{\boldsymbol{\theta}'} L_{\boldsymbol{\theta}}(\boldsymbol{\theta}') = \partial_{\boldsymbol{\theta}} J(\boldsymbol{\theta})$. Obviously, a reward which precedes an action in an rollout, can neither be caused by the action nor cause an action in the same rollout. Thus, when inserting Equations (2) into Equation (5), all cross-products between $r_t$ and $\partial_{\boldsymbol{\theta}} \log \pi(\mathbf{a}_{t+\delta t}|\mathbf{s}_{t+\delta t}, t+\delta t)$ for $\delta t > 0$ become zero in expectation [10]. Therefore, we can omit these terms and rewrite the estimator as

$$\partial_{\boldsymbol{\theta}'} L_{\boldsymbol{\theta}}(\boldsymbol{\theta}') = E\left\{\sum_{t=1}^{T} \partial_{\boldsymbol{\theta}'} \log \pi(\mathbf{a}_t|\mathbf{s}_t, t) Q^{\pi}(\mathbf{s}, \mathbf{a}, t)\right\}, \quad (6)$$

where $Q^{\pi}(\mathbf{s}, \mathbf{a}, t) = E\{\sum_{\tilde{t}=t}^{T} r(\mathbf{s}_{\tilde{t}}, \mathbf{a}_{\tilde{t}}, \mathbf{s}_{\tilde{t}+1}, \tilde{t})|\mathbf{s}_t = \mathbf{s}, \mathbf{a}_t = \mathbf{a}\}$ is called the state-action value function [1]. Equation (6) is equivalent to the policy gradient theorem [8] for $\boldsymbol{\theta}' \to \boldsymbol{\theta}$ in the infinite horizon case where the dependence on time $t$ can be dropped.

The derivation results in the Natural Actor Critic as discussed in [9, 10] when adding an additional punishment to prevent large steps away from the observed path distribution. This can be achieved by restricting the amount of change in the path distribution and, subsequently, determining the steepest descent for a fixed step away from the observed trajectories. Change in probability distributions is naturally measured using the Kullback-Leibler divergence, thus, after adding the additional constraint of $D(p_{\boldsymbol{\theta}'}(\tau)\|p_{\boldsymbol{\theta}}(\tau)) \approx 0.5(\boldsymbol{\theta}' - \boldsymbol{\theta})^{\mathrm{T}} \mathbf{F}(\boldsymbol{\theta})(\boldsymbol{\theta}' - \boldsymbol{\theta}) = \delta$ using a second-order expansion as approximation where $\mathbf{F}(\boldsymbol{\theta})$ denotes the Fisher information matrix [9, 10].

**Policy Search via Expectation Maximization.** One major drawback of gradient-based approaches is the learning rate, an open parameter which can be hard to tune in control problems but is essential for good performance. Expectation-Maximization algorithms are well-known to avoid this problem in supervised learning while even yielding faster convergence [16]. Previously, similar ideas have been explored in immediate reinforcement learning [17, 18]. In general, an EM-algorithm would choose the next policy parameters $\boldsymbol{\theta}_{n+1}$ such that $\boldsymbol{\theta}_{n+1} = \text{argmax}_{\boldsymbol{\theta}'} L_{\boldsymbol{\theta}}(\boldsymbol{\theta}')$. In the case where $\pi(\mathbf{a}_t|\mathbf{s}_t, t)$ belongs to the exponential family, the next policy can be determined analytically by setting Equation (6) to zero, i.e.,

$$E\left\{\sum_{t=1}^{T} \partial_{\boldsymbol{\theta}'} \log \pi(\mathbf{a}_t|\mathbf{s}_t, t) Q^{\pi}(\mathbf{s}, \mathbf{a}, t)\right\} = 0, \quad (7)$$

---

**Algorithm 1** Policy learning by Weighting Exploration with the Returns for Motor Primitives

**Input:** initial policy parameters $\boldsymbol{\theta}_0$
**repeat**
    *Sample*: Perform rollout(s) using $\mathbf{a} = (\boldsymbol{\theta} + \boldsymbol{\varepsilon}_t)^{\mathrm{T}} \boldsymbol{\phi}(\mathbf{s}, t)$ with $[\boldsymbol{\varepsilon}_t]_{ij} \sim \mathcal{N}(0, \sigma_{ij}^2)$ as stochastic policy and collect all $(t, \mathbf{s}_t, \mathbf{a}_t, \mathbf{s}_{t+1}, \boldsymbol{\varepsilon}_t, r_{t+1})$ for $t = \{1, 2, \ldots, T+1\}$.
    *Estimate*: Use unbiased estimate $\hat{Q}^\pi(\mathbf{s}, \mathbf{a}, t) = \sum_{\tilde{t}=t}^T r(\mathbf{s}_{\tilde{t}}, \mathbf{a}_{\tilde{t}}, \mathbf{s}_{\tilde{t}+1}, \tilde{t})$.
    *Reweight*: Compute importance weights and reweight rollouts, discard low-importance rollouts.
    *Update* policy using $\boldsymbol{\theta}_{k+1} = \boldsymbol{\theta}_k + \left\langle \sum_{t=1}^T \boldsymbol{\varepsilon}_t Q^\pi(\mathbf{s}, \mathbf{a}, t) \right\rangle_{w(\boldsymbol{\tau})} \Big/ \left\langle \sum_{t=1}^T Q^\pi(\mathbf{s}, \mathbf{a}, t) \right\rangle_{w(\boldsymbol{\tau})}$.
**until** Convergence $\boldsymbol{\theta}_{k+1} \approx \boldsymbol{\theta}_k$

---

and solving for $\boldsymbol{\theta}'$. Depending on the choice of a stochastic policy, we will obtain different solutions and different learning algorithms. It allows the extension of the reward-weighted regression to larger horizons as well as the introduction of the Policy learning by Weighting Exploration with the Returns (PoWER) algorithm.

### 2.3 Policy learning by Weighting Exploration with the Returns (PoWER)

In most learning control problems, we attempt to have a deterministic mean policy $\bar{\mathbf{a}} = \boldsymbol{\theta}^{\mathrm{T}} \boldsymbol{\phi}(\mathbf{s}, t)$ with parameters $\boldsymbol{\theta}$ and basis functions $\boldsymbol{\phi}$. In Section 3, we will introduce the basis functions of the motor primitives. When learning motor primitives, we turn this deterministic mean policy $\bar{\mathbf{a}} = \boldsymbol{\theta}^{\mathrm{T}} \boldsymbol{\phi}(\mathbf{s}, t)$ into a stochastic policy using additive exploration $\boldsymbol{\varepsilon}(\mathbf{s}, t)$ in order to make model-free reinforcement learning possible, i.e., we always intend to have a policy $\pi(\mathbf{a}_t | \mathbf{s}_t, t)$ which can be brought into the form $\mathbf{a} = \boldsymbol{\theta}^{\mathrm{T}} \boldsymbol{\phi}(\mathbf{s}, t) + \boldsymbol{\epsilon}(\boldsymbol{\phi}(\mathbf{s}, t))$. Previous work in this context [7, 4, 10, 18], with the notable exception of [19], has focused on state-independent, white Gaussian exploration, i.e., $\boldsymbol{\epsilon}(\boldsymbol{\phi}(\mathbf{s}, t)) \sim \mathcal{N}(0, \Sigma)$. It is straightforward to obtain the Reward-Weighted Regression for episodic RL by solving Equation (7) for $\boldsymbol{\theta}'$ which naturally yields a weighted regression method with the state-action values $Q^\pi(\mathbf{s}, \mathbf{a}, t)$ as weights. This form of exploration has resulted into various applications in robotics such as T-Ball batting, Peg-In-Hole, humanoid robot locomotion, constrained reaching movements and operational space control, see [4, 10, 18] for both reviews and their own applications.

However, such unstructured exploration at every step has a multitude of disadvantages: it causes a large variance which grows with the number of time-steps [19, 10], it perturbs actions too frequently 'washing' out their effects and can damage the system executing the trajectory. As a result, all methods relying on this state-independent exploration have proven too fragile for learning the Ball-in-a-Cup task on a real robot system. Alternatively, as introduced by [19], one could generate a form of structured, state-dependent exploration $\boldsymbol{\epsilon}(\boldsymbol{\phi}(\mathbf{s}, t)) = \boldsymbol{\varepsilon}_t^{\mathrm{T}} \boldsymbol{\phi}(\mathbf{s}, t)$ with $[\boldsymbol{\varepsilon}_t]_{ij} \sim \mathcal{N}(0, \sigma_{ij}^2)$, where $\sigma_{ij}^2$ are meta-parameters of the exploration that can also be optimized. This argument results into the policy $\mathbf{a} \sim \pi(\mathbf{a}_t | \mathbf{s}_t, t) = \mathcal{N}(\mathbf{a} | \boldsymbol{\theta}^{\mathrm{T}} \boldsymbol{\phi}(\mathbf{s}, t), \hat{\boldsymbol{\Sigma}}(\mathbf{s}, t))$. Inserting the resulting policy into Equation (7), we obtain the optimality condition in the sense of Equation (7) and can derive the update rule

$$\boldsymbol{\theta}' = \boldsymbol{\theta} + E\left\{ \sum_{t=1}^T Q^\pi(\mathbf{s}, \mathbf{a}, t) \mathbf{W}(\mathbf{s}, t) \right\}^{-1} E\left\{ \sum_{t=1}^T Q^\pi(\mathbf{s}, \mathbf{a}, t) \mathbf{W}(\mathbf{s}, t) \boldsymbol{\varepsilon}_t \right\} \qquad (8)$$

with $\mathbf{W}(\mathbf{s}, t) = \boldsymbol{\phi}(\mathbf{s}, t) \boldsymbol{\phi}(\mathbf{s}, t)^{\mathrm{T}} / (\boldsymbol{\phi}(\mathbf{s}, t)^{\mathrm{T}} \boldsymbol{\phi}(\mathbf{s}, t))$. Note that for our motor primitives $\mathbf{W}$ reduces to a diagonal, constant matrix and cancels out. Hence the simplified form in Algorithm 1. In order to reduce the number of rollouts in this on-policy scenario, we reuse the rollouts through importance sampling as described in the context of reinforcement learning in [1]. To avoid the fragility sometimes resulting from importance sampling in reinforcement learning, samples with very small importance weights are discarded. The expectations $E\{\cdot\}$ are replaced by the importance sampler denoted by $\langle \cdot \rangle_{w(\boldsymbol{\tau})}$. The resulting algorithm is shown in Algorithm 1. As we will see in Section 3, this PoWER method outperforms all other described methods significantly.

## 3 Application to Motor Primitive Learning for Robotics

In this section, we demonstrate the effectiveness of the algorithm presented in Section 2.3 in the context of motor primitive learning for robotics. For doing so, we will first give a quick overview how the motor primitives work and how the algorithm can be used to adapt them. As first evaluation, we will show that the novel presented PoWER algorithm outperforms many previous well-known

methods, i.e., 'Vanilla' Policy Gradients, Finite Difference Gradients, the Episodic Natural Actor Critic and the generalized Reward-Weighted Regression on the two simulated benchmark problems suggested in [10] and a simulated Underactuated Swing-Up [21]. Real robot applications are done with our best benchmarked method, the PoWER method. Here, we first show PoWER can learn the Underactuated Swing-Up [21] even on a real robot. As a significantly more complex motor learning task, we show how the robot can learn a high-speed Ball-in-a-Cup [24] movement with motor primitives for all seven degrees of freedom of our Barrett WAM™ robot arm.

## 3.1 Using the Motor Primitives in Policy Search

The motor primitive framework [22, 23] can be described as two coupled differential equations, i.e., we have a canonical system $\dot{\mathbf{y}} = \mathbf{f}(\mathbf{y}, \mathbf{z})$ with movement phase $\mathbf{y}$ and possible external coupling to $\mathbf{z}$ as well as a nonlinear system $\ddot{\mathbf{x}} = \mathbf{g}(\mathbf{x}, \dot{\mathbf{x}}, \mathbf{y}, \boldsymbol{\theta})$ which yields the current action for the system. Both dynamical systems are chosen to be stable and to have the right properties so that they are useful for the desired class of motor control problems. In this paper, we focus on single stroke movements as they frequently appear in human motor control [14, 23] and, thus, we will always choose the point attractor version of the motor primitives exactly as presented in [23] and not the older one in [22].

The biggest advantage of the motor primitive framework of [22, 23] is that the function $\mathbf{g}$ is linear in the policy parameters $\boldsymbol{\theta}$ and, thus, well-suited for imitation learning as well as for our presented reinforcement learning algorithm. For example, if we would have to learn only a motor primitive for a single degree of freedom $q_i$, then we could use a motor primitive in the form $\ddot{\bar{q}}_i = g(q_i, \dot{q}_i, y, \boldsymbol{\theta}) = \phi(\mathbf{s})^{\mathrm{T}} \boldsymbol{\theta}$ where $\mathbf{s} = [q_i, \dot{q}_i, y]$ is the state and where time is implicitly embedded in $y$. We use the output of $\ddot{\bar{q}}_i = \phi(\mathbf{s})^{\mathrm{T}} \boldsymbol{\theta} = \bar{a}$ as the policy mean. The perturbed accelerations $\ddot{q}_i = a = \bar{a} + \varepsilon$ is given to the system. The details of $\phi$ are given in [23].

In Sections 3.3 and 3.4, we use imitation learning for the initialization. For imitations, we follow [22]: first, extract the duration of the movement from initial and final zero velocity and use it to adjust the time constants. Second, use locally-weighted regression to solve for an imitation from a single example.

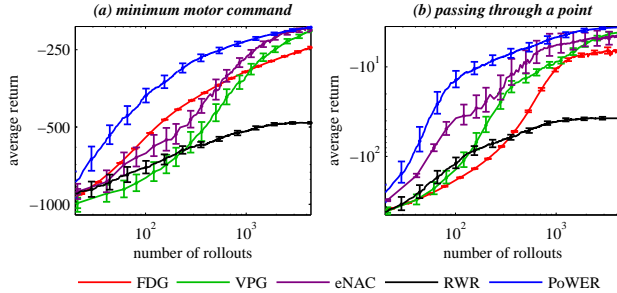

## 3.2 Benchmark Comparison

As benchmark comparison, we intend to follow a previously studied scenario in order to evaluate which method is best-suited for our problem class. For doing so, we perform our evaluations on the exact same benchmark problems as [10] and use two tasks commonly studied in mo-

Figure 1: This figure shows the mean performance of all compared methods in two benchmark tasks averaged over twenty learning runs with the error bars indicating the standard deviation. Policy learning by Weighting Exploration with the Returns (PoWER) clearly outperforms Finite Difference Gradients (FDG), 'Vanilla' Policy Gradients (VPG), the Episodic Natural Actor Critic (eNAC) and the adapted Reward-Weighted Regression (RWR) for both tasks.

tor control literature for which the analytic solutions are known, i.e., a reaching task where a goal has to be reached at a certain time while the used motor commands have to be minimized and a reaching task of the same style with an additional via-point. In this comparison, we mainly want to show the suitability of our algorithm and show that it outperforms previous methods such as Finite Difference Gradient (FDG) methods [10], 'Vanilla' Policy Gradients (VPG) with optimal baselines [7, 8, 11, 10], the Episodic Natural Actor Critic (eNAC) [9, 10], and the episodic version of the Reward-Weighted Regression (RWR) algorithm [18]. For both tasks, we use the same rewards as in [10] but we use the newer form of the motor primitives from [23]. All open parameters were manually optimized for each algorithm in order to maximize the performance while not destabilizing the convergence of the learning process.

When applied in the episodic scenario, Policy learning by Weighting Exploration with the Returns (PoWER) clearly outperformed the Episodic Natural Actor Critic (eNAC), 'Vanilla' Policy Gradient (VPG), Finite Difference Gradient (FDG) and the adapted Reward-Weighted Regression (RWR) for both tasks. The episodic Reward-Weighted Regression (RWR) is outperformed by all other algorithms suggesting that this algorithm does not generalize well from the immediate reward case.

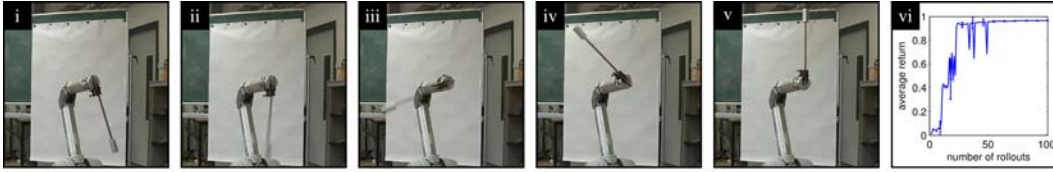

Figure 2: This figure shows the time series of the Underactuated Swing-Up where only a single joint of the robot is moved with a torque limit ensured by limiting the maximal motor current of that joint. The resulting motion requires the robot to (i) first move away from the target to limit the maximal required torque during the swing-up in (ii-iv) and subsequent stabilization (v). The performance of the PoWER method on the real robot is shown in (vi).

While FDG gets stuck on a plateau, both eNAC and VPG converge to the same, good final solution. PoWER finds the same (or even slightly better) solution while achieving it noticeably faster. The results are presented in Figure 1. Note that this plot has logarithmic scales on both axes, thus a unit difference corresponds to an order of magnitude. The omission of the first twenty rollouts was necessary to cope with the log-log presentation.

### 3.3 Underactuated Swing-Up

As additional simulated benchmark and for the real-robot evaluations, we employed the Underactuated Swing-Up [21]. Here, only a single degree of freedom is represented by the motor primitive as described in Section 3.1. The goal is to move a hanging heavy pendulum to an upright position and stabilize it there in minimum time and with minimal motor torques. By limiting the motor current for that degree of freedom, we can ensure that the torque limits described in [21] are maintained and directly moving the joint to the right position is not possible. Under these torque limits, the robot needs to (i) first move away from the target to limit the maximal required torque during the swing-up in (ii-iv) and subsequent stabilization (v) as illustrated in Figure 2 (i-v). This problem is similar to a mountain-car problem where the car would have to stop on top or experience a failure.

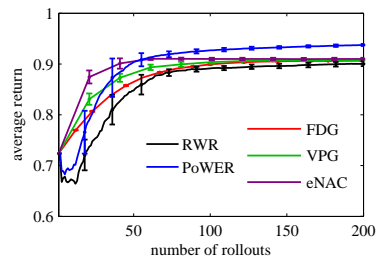

Figure 3: This figure shows the performance of all compared methods for the swing-up in simulation and show the mean performance averaged over 20 learning runs with the error bars indicating the standard deviation. PoWER outperforms the other algorithms from 50 rollouts on and finds a significantly better policy.

The applied torque limits were the same as in [21] and so was the reward function was the except that the complete return of the trajectory was transformed by an $\exp(\cdot)$ to ensure positivity. Again all open parameters were manually optimized. The motor primitive with nine shape parameters and one goal parameter was initialized by imitation learning from a kinesthetic teach-in. Subsequently, we compared the other algorithms as previously considered in Section 3.2 and could show that PoWER would again outperform them. The results are given in Figure 3. As it turned out to be the best performing method, we then used it successfully for learning optimal swing-ups on a real robot. See Figure 2 (vi) for the resulting real-robot performance.

### 3.4 Ball-in-a-Cup on a Barrett WAM™

The most challenging application in this paper is the children's game Ball-in-a-Cup [24] where a small cup is attached at the robot's end-effector and this cup has a small wooden ball hanging down from the cup on a 40cm string. Initially, the ball is hanging down vertically. The robot needs to move fast in order to induce a motion at the ball through the string, swing it up and catch it with the cup, a possible movement is illustrated in Figure 4 (top row). The state of the system is described in joint angles and velocities of the robot and the Cartesian coordinates of the ball. The actions are the joint space accelerations where each of the seven joints is represented by a motor primitive. All motor primitives are perturbed separately but employ the same joint final reward given by $r(t_c) = \exp(-\alpha(x_c - x_b)^2 - \alpha(y_c - y_b)^2)$ while $r(t) = 0$ for all other $t \neq t_c$ where $t_c$ is the moment where the ball passes the rim of the cup with a downward direction, the cup position denoted by $[x_c, y_c, z_c] \in \mathbb{R}^3$, the ball position $[x_b, y_b, z_b] \in \mathbb{R}^3$ and a scaling parameter $\alpha = 100$. The task is quite complex as the reward is not modified solely by the movements of the cup but foremost by the movements of the ball and the movements of the ball are very sensitive to changes in the movement. A small perturbation of the initial condition or during the trajectory will drastically change the movement of the ball and hence the outcome of the rollout.

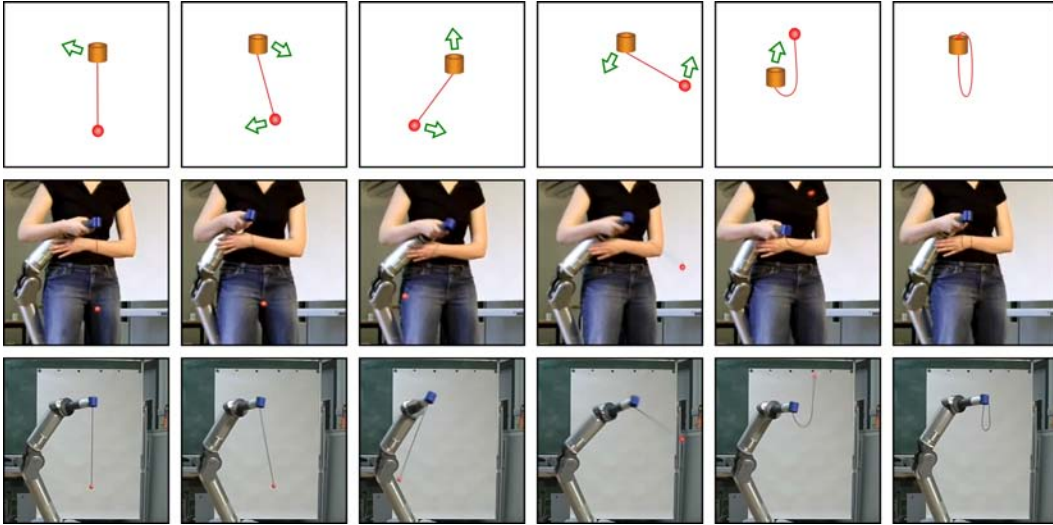

Figure 4: This figure shows schematic drawings of the Ball-in-a-Cup motion, the final learned robot motion as well as a kinesthetic teach-in. The green arrows show the directions of the current movements in that frame. The human cup motion was taught to the robot by imitation learning with 31 parameters per joint for an approximately 3 seconds long trajectory. The robot manages to reproduce the imitated motion quite accurately, but the ball misses the cup by several centimeters. After ca. 75 iterations of our Policy learning by Weighting Exploration with the Returns (PoWER) algorithm the robot has improved its motion so that the ball goes in the cup. Also see Figure 5.

Due to the complexity of the task, Ball-in-a-Cup is even a hard motor learning task for children who usually only succeed at it by observing another person playing and a lot of improvement by trial-and-error. Mimicking how children learn to play Ball-in-a-Cup, we first initialize the motor primitives by imitation and, subsequently, improve them by reinforcement learning. We recorded the motions of a human player by kinesthetic teach-in in order to obtain an example for imitation as shown in Figure 4 (middle row). From the imitation, it can be determined by cross-validation that 31 parameters per motor primitive are needed. As expected, the robot fails to reproduce the the presented

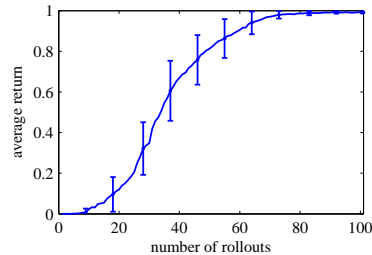

Figure 5: This figure shows the expected return of the learned policy in the Ball-in-a-Cup evaluation averaged over 20 runs.

behavior and reinforcement learning is needed for self-improvement. Figure 5 shows the expected return over the number of rollouts where convergence to a maximum is clearly recognizable. The robot regularly succeeds at bringing the ball into the cup after approximately 75 iterations.

## 4 Conclusion

In this paper, we have presented a new perspective on policy learning methods and an application to a highly complex motor learning task on a real Barrett WAM™ robot arm. We have generalized the previous work in [17, 18] from the immediate reward case to the episodic case. In the process, we could show that policy gradient methods are a special case of this more general framework. During initial experiments, we realized that the form of exploration highly influences the speed of the policy learning method. This empirical insight resulted in a novel policy learning algorithm, Policy learning by Weighting Exploration with the Returns (PoWER), an EM-inspired algorithm that outperforms several other policy search methods both on standard benchmarks as well as on a simulated Underactuated Swing-Up.

We successfully applied this novel PoWER algorithm in the context of learning two tasks on a physical robot, i.e., the Underacted Swing-Up and Ball-in-a-Cup. Due to the curse of dimensionality, we cannot start with an arbitrary solution. Instead, we mimic the way children learn Ball-in-a-Cup and first present an example for imitation learning which is recorded using kinesthetic teach-in. Subsequently, our reinforcement learning algorithm takes over and learns how to move the ball into

the cup reliably. After only realistically few episodes, the task can be regularly fulfilled and the robot shows very good average performance.

# References

[1] R. Sutton and A. Barto. *Reinforcement Learning*. MIT Press, 1998.

[2] J. Bagnell, S. Kadade, A. Ng, and J. Schneider. Policy search by dynamic programming. In *Advances in Neural Information Processing Systems (NIPS)*, 2003.

[3] A. Ng and M. Jordan. PEGASUS: A policy search method for large MDPs and POMDPs. In *International Conference on Uncertainty in Artificial Intelligence (UAI)*, 2000.

[4] F. Guenter, M. Hersch, S. Calinon, and A. Billard. Reinforcement learning for imitating constrained reaching movements. *RSJ Advanced Robotics*, 21, 1521-1544, 2007.

[5] M. Toussaint and C. Goerick. Probabilistic inference for structured planning in robotics. In *International Conference on Intelligent Robots and Systems (IROS)*, 2007.

[6] M. Hoffman, A. Doucet, N. de Freitas, and A. Jasra. Bayesian policy learning with transdimensional MCMC. In *Advances in Neural Information Processing Systems (NIPS)*, 2007.

[7] R. J. Williams. Simple statistical gradient-following algorithms for connectionist reinforcement learning. *Machine Learning*, 8:229–256, 1992.

[8] R. S. Sutton, D. McAllester, S. Singh, and Y. Mansour. Policy gradient methods for reinforcement learning with function approximation. In *Advances in Neural Information Processing Systems (NIPS)*, 2000.

[9] J. Bagnell and J. Schneider. Covariant policy search. In *International Joint Conference on Artificial Intelligence (IJCAI)*, 2003.

[10] J. Peters and S. Schaal. Policy gradient methods for robotics. In *International Conference on Intelligent Robots and Systems (IROS)*, 2006.

[11] G. Lawrence, N. Cowan, and S. Russell. Efficient gradient estimation for motor control learning. In *International Conference on Uncertainty in Artificial Intelligence (UAI)*, 2003.

[12] H. Attias. Planning by probabilistic inference. In *Ninth International Workshop on Artificial Intelligence and Statistics (AISTATS)*, 2003.

[13] J. Binder, D. Koller, S. Russell, and K. Kanazawa. Adaptive probabilistic networks with hidden variables. *Machine Learning*, 29:213–244, 1997.

[14] G. Wulf. *Attention and motor skill learning*. Human Kinetics, Champaign, IL, 2007.

[15] D. E. Kirk. *Optimal control theory*. Prentice-Hall, Englewood Cliffs, New Jersey, 1970.

[16] G. J. McLachan and T. Krishnan. *The EM Algorithm and Extensions*. Wiley Series in Probability and Statistics. John Wiley & Sons, 1997.

[17] P. Dayan and G. E. Hinton. Using expectation-maximization for reinforcement learning. *Neural Computation*, 9(2):271–278, 1997.

[18] J. Peters and S. Schaal. Reinforcement learning by reward-weighted regression for operational space control. In *International Conference on Machine Learning (ICML)*, 2007.

[19] T. Rückstieß, M. Felder, and J. Schmidhuber. State-dependent exploration for policy gradient methods. In *European Conference on Machine Learning (ECML)*, 2008.

[20] M. Kawato, F. Gandolfo, H. Gomi, and Y. Wada. Teaching by showing in kendama based on optimization principle. In *International Conference on Artificial Neural Networks*, 1994.

[21] C. G. Atkeson. Using local trajectory optimizers to speed up global optimization in dynamic programming. In *Advances in Neural Information Processing Systems (NIPS)*, 1994.

[22] A. Ijspeert, J. Nakanishi, and S. Schaal. Learning attractor landscapes for learning motor primitives. In *Advances in Neural Information Processing Systems (NIPS)*, 2003.

[23] S. Schaal, P. Mohajerian, and A. Ijspeert. Dynamics systems vs. optimal control — a unifying view. *Progress in Brain Research*, 165(1):425–445, 2007.

[24] Wikipedia, May 31, 2008. `http://en.wikipedia.org/wiki/Ball_in_a_cup`

[25] J. Kober, B. Mohler, and J. Peters. Learning perceptual coupling for motor primitives. In *International Conference on Intelligent RObots and Systems (IROS)*, 2008.

